# Non-stationary continuous dynamic Bayesian networks

**Marco Grzegorczyk**
Department of Statistics, TU Dortmund University, 44221 Dortmund, Germany
grzegorczyk@statistik.tu-dortmund.de

**Dirk Husmeier**
Biomathematics & Statistics Scotland (BioSS)
JCMB, The King's Buildings, Edinburgh EH93JZ, United Kingdom
dirk@bioss.ac.uk

## Abstract

Dynamic Bayesian networks have been applied widely to reconstruct the structure of regulatory processes from time series data. The standard approach is based on the assumption of a homogeneous Markov chain, which is not valid in many real-world scenarios. Recent research efforts addressing this shortcoming have considered undirected graphs, directed graphs for discretized data, or over-flexible models that lack any information sharing among time series segments. In the present article, we propose a non-stationary dynamic Bayesian network for continuous data, in which parameters are allowed to vary among segments, and in which a common network structure provides essential information sharing across segments. Our model is based on a Bayesian multiple change-point process, where the number and location of the change-points is sampled from the posterior distribution.

## 1  Introduction

There has recently been considerable interest in structure learning of Bayesian networks. Examples from the topical field of systems biology are the reconstruction of transcriptional regulatory networks from gene expression data [1], the inference of signal transduction pathways from protein concentrations [2], and the identification of neural information flow operating in the brains of songbirds [3]. In particular, dynamic Bayesian networks (DBNs) have been applied, as they allow feedback loops and recurrent regulatory structures to be modelled while avoiding the ambiguity about edge directions common to static Bayesian networks. The standard assumption underpinning DBNs is that of stationarity: time-series data are assumed to have been generated from a homogeneous Markov process. However, regulatory interactions and signal transduction processes in the cell are usually adaptive and change in response to external stimuli. Likewise, neural information flow slowly adapts via Hebbian learning to make the processing of sensory information more efficient. The assumption of stationarity is therefore too restrictive in many circumstances, and can potentially lead to erroneous conclusions.

In the recent past, various research efforts have addressed this issue and proposed models that relax the stationarity assumption. Talih and Hengartner [4] proposed a time-varying Gaussian graphical model (GGM), in which the time-varying variance structure of the data was inferred with reversible jump (RJ) Markov chain Monte Carlo (MCMC). A limitation of this approach is that changes of the network structure between different segments are restricted to changing at most a single edge, and the total number of segments is assumed known a priori. Xuan and Murphy [5] developed a related non-stationary GGM based on a product partition model. The method allows for separate structures

|  | Proposed here | Robinson & Hartemink (2009) | Lèbre (2008) | Grzegorcyk et al. (2008) | Ko et al. (2007) |
|---|---|---|---|---|---|
| Score | Marginal Likelihood | Marginal Likelihood | Marginal Likelihood | Marginal Likelihood | BIC |
| Change-points | node specific | whole network | node specific | whole network | node specific |
| Structure constant | Yes | No | No | Yes | Yes |
| Data format | Continuous | Discrete | Continuous | Continuous | Continuous |
| Latent variables | Change-point process | Change-point process | Change-point process | Free allocation | Free allocation |

Table 1: Overview of how our model compares with various related, recently published models.

in different segments, where the number of structures is inferred from the data. The inference algorithm iterates between a convex optimization for determining the graph structure and a dynamic programming algorithm for calculating the segmentation. The latter aspect imposes restrictions on the graph structure (decomposability), though. Moreover, both the models of [4] and [5] are based on undirected graphs, whereas most processes in systems biology, like neural information flow, signal transduction and transcriptional regulation, are intrinsically of a directed nature. To address this shortcoming, Robinson and Hartemink [6] and Lébre [7] proposed a non-stationary dynamic Bayesian network. Both methods allow for different network structures in different segments of the time series, where the location of the change-points and the total number of segments are inferred from the data with RJMCMC. The essential difference between the two methods is that the model proposed in [6] is a non-stationary version of the BDe score [8], which requires the data to be discretized. The method proposed in [7] is based on the Bayesian linear regression model of [9], which avoids the need for data discretization.

Allowing the network structure to change between segments leads to a highly flexible model. However, this approach faces a conceptual and a practical problem. The *practical* problem is potential model over-flexibility[1]. Owing to the high costs of postgenomic high-throughput experiments, time series in systems biology are typically rather short. Modelling short time series segments with separate network structures will almost inevitably lead to inflated inference uncertainty, which calls for some information sharing between the segments. The *conceptual* problem is related to the very premise of a flexible network structure. This assumption is reasonable for some scenarios, like morphogenesis, where the different segments are e.g. associated with the embryonic, larval, pupal, and adult stages of fruit fly (as discussed in [6]). However, for most cellular processes on a shorter time scale, it is questionable whether it is the structure rather than just the strength of the regulatory interactions that changes with time. To use the analogy of the traffic flow network invoked in [6]: it is not the road system (the network structure) that changes between off-peak and rush hours, but the intensity of the traffic flow (the strength of the interactions). In the same vein, it is not the ability of a transcription factor to potentially bind to the promoter of a gene and thereby initiate transcription (the interaction structure), but the extent to which this happens (the interaction strength).

The objective of the present work is to propose and assess a non-stationary continuous-valued DBN that introduces information sharing among different time series segments via a constrained structure. Our model is non-stationary with respect to the parameters, while the network structure is kept fixed among segments. Our model complements the one proposed in [6] in two other aspects: the score is a non-stationary generalization of the BGe [10] rather than the BDe score, thus avoiding the need for data discretization, and the patterns of non-stationarity are node-specific, thereby providing extra model flexibility. Our work is based on [11], [12], and [13]. Like [11], our model is effectively a mixture of BGe models. We replace the free allocation model of [11] by a change-point process to incorporate our prior notion that adjacent time points in a time series are likely to be governed by similar distributions. We borrow from [12] the concept of node-specific change-points to enable greater model flexibility. However, as opposed to [12], we do not approximate the scoring function by BIC [14], but compute the proper marginal likelihood. The objective of inference is to infer the

location and the node-specific number of change-points from the posterior distribution. An overview of how our method is related to various recently published related models is provided in Table 1.

## 2 Methodology

### 2.1 The dynamic BGe network

DBNs are flexible models for representing probabilistic relationships between interacting variables (nodes) $X_1, \ldots, X_N$ via a directed graph $\mathcal{G}$. An edge pointing from $X_i$ to $X_j$ indicates that the realization of $X_j$ at time point $t$, symbolically: $X_j(t)$, is conditionally dependent on the realization of $X_i$ at time point $t-1$, symbolically: $X_i(t-1)$. The parent node set of node $X_n$ in $\mathcal{G}$, $\pi_n = \pi_n(\mathcal{G})$, is the set of all nodes from which an edge points to node $X_n$ in $\mathcal{G}$. Given a data set $\mathcal{D}$, where $\mathcal{D}_{n,t}$ and $\mathcal{D}_{(\pi_n,t)}$ are the $t$th realizations $X_n(t)$ and $\pi_n(t)$ of $X_n$ and $\pi_n$, respectively, and $1 \leq t \leq m$ represents time, DBNs are based on the following homogeneous Markov chain expansion:

$$P(\mathcal{D}|\mathcal{G}, \boldsymbol{\theta}) = \prod_{n=1}^{N} \prod_{t=2}^{m} P\Big(X_n(t) = \mathcal{D}_{n,t} | \pi_n(t-1) = \mathcal{D}_{(\pi_n,t-1)}, \boldsymbol{\theta}_n\Big) \tag{1}$$

where $\boldsymbol{\theta}$ is the total parameter vector, composed of node-specific subvectors $\boldsymbol{\theta}_n$, which specify the local conditional distributions in the factorization. From Eq. (1) and under the assumption of parameter independence, $P(\boldsymbol{\theta}|\mathcal{G}) = \prod_n P(\boldsymbol{\theta}_n|\mathcal{G})$, the marginal likelihood is given by

$$P(\mathcal{D}|\mathcal{G}) = \int P(\mathcal{D}|\mathcal{G}, \boldsymbol{\theta}) P(\boldsymbol{\theta}|\mathcal{G}) d\boldsymbol{\theta} = \prod_{n=1}^{N} \Psi(\mathcal{D}_n^{\pi_n}, \mathcal{G}) \tag{2}$$

$$\Psi(\mathcal{D}_n^{\pi_n}, \mathcal{G}) = \int \prod_{t=2}^{m} P\Big(X_n(t) = \mathcal{D}_{n,t} | \pi_n(t-1) = \mathcal{D}_{(\pi_n,t-1)}, \boldsymbol{\theta}_n\Big) P(\boldsymbol{\theta}_n|\mathcal{G}) d\boldsymbol{\theta}_n \tag{3}$$

where $\mathcal{D}_n^{\pi_n} := \{(\mathcal{D}_{n,t}, \mathcal{D}_{\pi_n,t-1}) : 2 \leq t \leq m\}$ is the subset of data pertaining to node $X_n$ and parent set $\pi_n$. We choose a linear Gaussian distribution for the local conditional distribution $P(X_n|\pi_n, \boldsymbol{\theta}_n)$ in Eq.(1). Under fairly weak regularity conditions discussed in [10] (parameter modularity and conjugacy of the prior[2]), the integral in Eq. (3) has a closed form solution, given by Eq. (24) in [10]. The resulting expression is called the BGe score[3].

### 2.2 The non-stationary dynamic change-point BGe model (cpBGe)

To obtain a non-stationary DBN, we generalize Eq. (1) with a node-specific mixture model:

$$P(\mathcal{D}|\mathcal{G}, \mathbf{V}, \mathbf{K}, \boldsymbol{\theta}) = \prod_{n=1}^{N} \prod_{t=2}^{m} \prod_{k=1}^{\mathcal{K}_n} P\Big(X_n(t) = \mathcal{D}_{n,t} | \pi_n(t-1) = \mathcal{D}_{(\pi_n,t-1)}, \boldsymbol{\theta}_n^k\Big)^{\delta_{V_n(t),k}} \tag{4}$$

where $\delta_{V_n(t),k}$ is the Kronecker delta, $\mathbf{V}$ is a matrix of latent variables $V_n(t)$, $V_n(t) = k$ indicates that the realization of node $X_n$ at time $t$, $X_n(t)$, has been generated by the $k$th component of a mixture with $\mathcal{K}_n$ components, and $\mathbf{K} = (\mathcal{K}_1, \ldots, \mathcal{K}_n)$. Note that the matrix $\mathbf{V}$ divides the data into several disjoined subsets, each of which can be regarded as pertaining to a separate BGe model with parameters $\boldsymbol{\theta}_n^k$. The vectors $\mathbf{V}_n$ are node-specific, i.e. different nodes can have different breakpoints. The probability model defined in Eq.(4) is effectively a mixture model with local probability distributions $P(X_n|\pi_n, \boldsymbol{\theta}_n^k)$ and it can hence, under a free allocation of the latent variables, approximate any probability distribution arbitrarily closely. In the present work, we change the assignment of data points to mixture components from a free allocation to a change-point process. This effectively reduces the complexity of the latent variable space and incorporates our prior belief that, in a

time series, adjacent time points are likely to be assigned to the same component. From Eq. (4), the marginal likelihood conditional on the latent variables $\mathbf{V}$ is given by

$$P(\mathcal{D}|\mathcal{G},\mathbf{V},\mathbf{K})=\int P(\mathcal{D}|\mathcal{G},\mathbf{V},\mathbf{K},\boldsymbol{\theta})P(\boldsymbol{\theta})d\boldsymbol{\theta} \;\;=\;\; \prod_{n=1}^{N}\prod_{k=1}^{\mathcal{K}_n}\Psi(\mathcal{D}_n^{\pi_n}[k,\mathbf{V}_n],\mathcal{G}) \tag{5}$$

$$\Psi(\mathcal{D}_n^{\pi_n}[k,\mathbf{V}_n],\mathcal{G})=\int \prod_{t=2}^{m} P\Big(X_n(t)=\mathcal{D}_{n,t}|\pi_n(t-1)=\mathcal{D}_{(\pi_n,t-1)},\boldsymbol{\theta}_n^k\Big)^{\delta_{V_n(t),k}} P(\boldsymbol{\theta}_n^k|\mathcal{G})d\boldsymbol{\theta}_n^k \tag{6}$$

Eq. (6) is similar to Eq. (3), except that it is restricted to the subset $\mathcal{D}_n^{\pi_n}[k,\mathbf{V}_n] \;:=\; \{(\mathcal{D}_{n,t},\mathcal{D}_{\pi_n,t-1}) \,:\, V_n(t)=k, 2 \leq t \leq m\}$. Hence when the regularity conditions defined in [10] are satisfied, then the expression in Eq.(6) has a closed-form solution: it is given by Eq. (24) in [10] restricted to the subset of the data that has been assigned to the $k$th mixture component (or $k$th segment). The joint probability distribution of the proposed cpBGe model is given by:

$$
\begin{aligned}
P(\mathcal{G},\mathbf{V},\mathbf{K},\mathcal{D}) \;\;&=\;\; P(\mathcal{D}|\mathcal{G},\mathbf{V},\mathbf{K})\cdot P(\mathcal{G})\cdot P(\mathbf{V}|\mathbf{K})\cdot P(\mathbf{K}) \\
&=\;\; P(\mathcal{G})\cdot \prod_{n=1}^{N}\left\{ \{P(\mathbf{V}_n|\mathcal{K}_n)\cdot P(\mathcal{K}_n)\cdot \prod_{k=1}^{\mathcal{K}_n}\Psi(\mathcal{D}_n^{\pi_n}[k,\mathbf{V}_n],\mathcal{G}) \right\}
\end{aligned}
\tag{7}
$$

In the absence of genuine prior knowledge about the regulatory network structure, we assume for $P(\mathcal{G})$ a uniform distribution on graphs, subject to a fan-in restriction of $|\pi_n| \leq 3$. As prior probability distributions on the node-specific numbers of mixture components $\mathcal{K}_n$, $P(\mathcal{K}_n)$, we take iid truncated Poisson distributions with shape parameter $\lambda = 1$, restricted to $1 \leq \mathcal{K}_n \leq \mathcal{K}_{MAX}$ (we set $\mathcal{K}_{MAX} = 10$ in our simulations). The prior distribution on the latent variable vectors, $P(\mathbf{V}|\mathbf{K}) = \prod_{n=1}^{N}\{P(\mathbf{V}_n|\mathcal{K}_n)$, is implicitly defined via the change-point process as follows. We identify $\mathcal{K}_n$ with $\mathcal{K}_n - 1$ change-points $\mathbf{b}_n = \{b_{n,1},\dots,b_{n,\mathcal{K}_n-1}\}$ on the continuous interval $[2,m]$. For notational convenience we introduce the pseudo change-points $b_{n,0}=2$ and $b_{n,\mathcal{K}_n}=m$. For node $X_n$ the observation at time point $t$ is assigned to the $k$th component, symbolically $V_n(t)=k$, if $b_{n,k-1} \leq t < b_{n,k}$. Following [15] we assume that the change-points are distributed as the even-numbered order statistics of $\mathcal{L} := 2(\mathcal{K}_n - 1) + 1$ points $u_1,\dots,u_{\mathcal{L}}$ uniformly and independently distributed on the interval $[2,m]$. The motivation for this prior, instead of taking $\mathcal{K}_n$ uniformly distributed points, is to encourage *a priori* an equal spacing between the change-points, i.e. to discourage mixture components (i.e. segments) that contain only a few observations. The even-numbered order statistics prior on the change-point locations $\mathbf{b}_n$ induces a prior distribution on the node-specific allocation vectors $\mathbf{V}_n$. Deriving a closed-form expression is involved. However, the MCMC scheme we discuss in the next section does not sample $\mathbf{V}_n$ directly, but is based on local modifications of $\mathbf{V}_n$ based on birth, death and reallocation moves. All that is required for the acceptance probabilities of these moves are $P(\mathbf{V}_n|\mathcal{K}_n)$ ratios, which are straightforward to compute.

## 2.3 MCMC inference

We now describe an MCMC algorithm to obtain a sample $\{\mathcal{G}^i,\mathbf{V}^i,\mathbf{K}^i\}_{i=1,\dots,I}$ from the posterior distribution $P(\mathcal{G},\mathbf{V},\mathbf{K}|\mathcal{D}) \propto P(\mathcal{G},\mathbf{V},\mathbf{K},\mathcal{D})$ of Eq. (7). We combine the structure MCMC algorithm[4] [17, 18] with the change-point model used in [15], and draw on the fact that conditional on the allocation vectors $\mathbf{V}$, the model parameters can be integrated out to obtain the marginal likelihood terms $\Psi(\mathcal{D}_n^{\pi_n}[k,\mathbf{V}_n],\mathcal{G})$ in closed form, as shown in the previous section. Note that this approach is equivalent to the idea underlying the allocation sampler proposed in [13]. The resulting algorithm is effectively an RJMCMC scheme [15] in the discrete space of network structures and latent allocation vectors, where the Jacobian in the acceptance criterion is always 1 and can be omitted. With probability $p_G = 0.5$ we perform a structure MCMC move on the current graph $\mathcal{G}^i$ and leave the latent variable matrix and the numbers of mixture components unchanged, symbolically: $\mathbf{V}^{i+1} = \mathbf{V}^i$ and $\mathbf{K}^{i+1} = \mathbf{K}^i$. A new candidate graph $\mathcal{G}^{i+1}$ is randomly drawn out of the set of graphs $\mathcal{N}(\mathcal{G}^i)$ that can be reached from the current graph $\mathcal{G}^i$ by deletion or addition of a single edge. The proposed graph $\mathcal{G}^{i+1}$ is accepted with probability:

$$A(\mathcal{G}^{i+1}|\mathcal{G}^i) = min\left\{1, \frac{P(\mathcal{D}|\mathcal{G}^{i+1},\mathbf{V}^i,\mathbf{K}^i)}{P(\mathcal{D}|\mathcal{G}^i,\mathbf{V}^i,\mathbf{K}^i)}\frac{P(\mathcal{G}^{i+1})}{P(\mathcal{G}^i)}\frac{|\mathcal{N}(\mathcal{G}^i)|}{|\mathcal{N}(\mathcal{G}^{i+1})|}\right\} \tag{8}$$

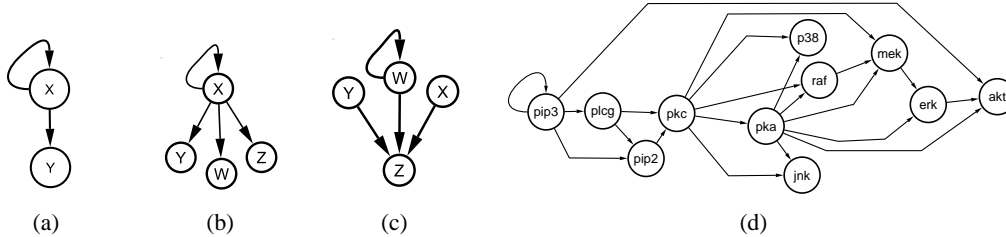

(a)          (b)          (c)          (d)

Figure 1: **Networks from which synthetic data were generated.** Panels (a-c) show elementary network motifs [20]. Panel (d) shows a protein signal transduction network studied in [2], with an added feedback loop on the root node.

where |.| is the cardinality, and the marginal likelihood terms have been specified in Eq. (5). The graph is left unchanged, symbolically $\mathcal{G}^{i+1} := \mathcal{G}^i$, if the move is not accepted.

With the complementary probability $1 - p_G$ we leave the graph $\mathcal{G}^i$ unchanged and perform a move on $(\mathbf{V}^i, \mathbf{K}^i)$, where $\mathbf{V}_n^i$ is the latent variable vector of $X_n$ in $\mathbf{V}^i$, and $\mathbf{K}^i = (\mathcal{K}_1^i, \dots, \mathcal{K}_N^i)$. We randomly select a node $X_n$ and change its current number of components $\mathcal{K}_n^i$ via a change-point birth or death move, or its latent variable vector $\mathbf{V}_n^i$ by a change-point re-allocation move. The change-point birth (death) move increases (decreases) $\mathcal{K}_n^i$ by 1 and may also have an effect on $\mathbf{V}_n^i$. The change-point reallocation move leaves $\mathcal{K}_n^i$ unchanged and may have an effect on $\mathbf{V}_n^i$. Under fairly mild regularity conditions (ergodicity), the MCMC sampling scheme converges to the desired posterior distribution if the acceptance probabilities for the three change-point moves $(\mathcal{K}_n^i, \mathbf{V}_n^i) \to (\mathcal{K}_n^{i+1}, \mathbf{V}_n^{i+1})$ are chosen of the form $min(1, R)$, see [15], with

$$R = \frac{\prod_{k=1}^{\mathcal{K}_n^{i+1}} \Psi(\mathcal{D}_n^{\pi_n}[k, \mathbf{V}_n^{i+1}], \mathcal{G})}{\prod_{k=1}^{K_n^i} \Psi(\mathcal{D}_n^{\pi_n}[k, \mathbf{V}_n^i], \mathcal{G})} \times A \times B \qquad (9)$$

where $A = P(\mathbf{V}_n^{i+1}|\mathcal{K}_n^{i+1})P(\mathcal{K}_n^{i+1})/P(\mathbf{V}_n^i|\mathcal{K}_n^i)P(\mathcal{K}_n^i)$ is the prior probability ratio, and $B$ is the inverse proposal probability ratio. The exact form of these factors depends on the move type and is provided in the supplementary material. We note that the implementation of the dynamic programming scheme proposed in [19] has the prospect to improve the convergence and mixing of the Markov chain, which we will investigate in our future work.

## 3 Results on synthetic data

To assess the performance of the proposed model, we applied it to a set of synthetic data generated from different networks, as shown in Figure 1. The structures in Figure panels 1a-c constitute elementary network motifs, as studied e.g. in [20]. The network in Figure 1d was extracted from the systems biology literature [2] and represents a well-studied protein signal transduction pathway. We added an extra feedback loop on the root node to allow the generation of a Markov chain with non-zero autocorrelation; note that this modification is not biologically implausible [21].

We generated data with a mixture of piece-wise linear processes and sinusoidal transfer functions. The advantage of the first approach is the exact knowledge of the true process change-points; the second approach is more realistic (smooth function) with a stronger mismatch between model and data-generation mechanism. For example, the network in Figure 1c was modelled as

$$X(t+1) = \phi_X(t); \quad Y(t+1) = \phi_Y(t); \quad W(t+1) = W(t) + \frac{2\pi}{m} + c_W \cdot \phi_W(t)$$
$$Z(t+1) = c_X \cdot X(t) + c_Y \cdot Y(t) + \cdot sin(W(t)) + c_Z \cdot \phi_Z(t+1) \qquad (10)$$

where the $\phi_{\cdot}(.)$ are iid standard Normally distributed. We employed different values $c_X = c_Y \in \{0.25, 0.5\}$ and $c_Z, c_W \in \{0.25, 0.5, 1\}$ to vary the signal-to-noise ratio and the amount of autocorrelation in $W$. For each parameter configuration, 25 time series with 41 time points where independently generated. For the other networks, data were generated in a similar way. Owing to space restrictions, the complete model specifications have to be relegated to the supplementary material.

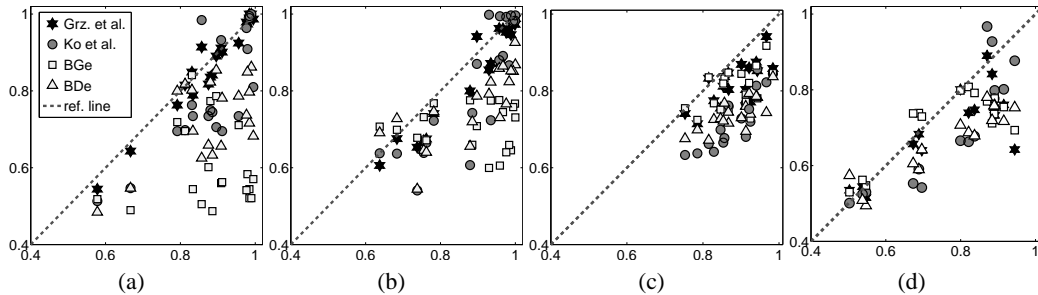

| $cpBGe$ vs. ... | (a) | (b) | (c) | (d) |
|---|---|---|---|---|
| ... vs. Grz. et al. | 0.753 | <0.0001 | <0.0001 | 0.013 |
| ... vs. Ko et al. | <0.0001 | 0.074 | <0.0001 | 0.002 |
| ... vs. $BGe$ | <0.0001 | <0.0001 | <0.0001 | 0.060 |
| ... vs. $BDe$ | <0.0001 | <0.0001 | <0.0001 | <0.0001 |

Figure 2: **Comparison of AUC scores on the synthetic data.** The panels (a-d) correspond to those of Figure 1. The horizontal axis in each panel represents the proposed cpBGe model. The vertical axis represents the following competing models: BDe ($\triangle$), BGe ($\square$), the method of Ko et al. [12] ($\bigcirc$), and the method of Grzegorczyk et al. [11] ($\star$), adapted as described in the text. Different symbols of the same shape correspond to different signal-to-noise ratios (SNR) and autocorrelation times (ACT). Each symbol shows a comparison of two average AUC scores, averaged over 25 (panels a-c) or 5 (panel d) time series independently generated for a given SNR/ACT setting. The diagonal line indicates equal performance; symbols below this lines indicate that the proposed cpBGe model outperforms the competing model. The table in the bottom shows an overview of the corresponding p-values obtained from a two-sided paired t-test with Bonferroni correction. For all but three cases the cpBGe model outperforms the competing model at the standard $5\%$ significance level.

To each data set, we applied the proposed cpBGe model as described in Section 2. We compared its performance with four alternative schemes. We chose the classical stationary DBNs based on BDe [8] and BGe [10]. Note that for these models the parameters can be integrated out analytically, and only the network structure has to be learned. The latter was sampled from the posterior distribution with structure MCMC [17, 18]. Note that the BDe model requires discretized data, which we effected with the information bottleneck algorithm [22]. Our comparative evaluation also included two non-linear/non-stationary models with a clearly defined network structure (for the sake of comparability with our approach). We chose the method of Ko et al. [12] for its flexibility and comparative ease of implementation. The inference scheme is based on the application of the EM algorithm [23] to a node-specific mixture model subject to a BIC penalty term [14]. We implemented this algorithm according to the authors' specification in MATLAB©, using the software package NETLAB [24]. We also compared our model with the approach proposed by Grzegorczyk et al. [11]. We applied the software available from the authors' website. We replaced the authors' free allocation model by the change-point process used for our model. This was motivated by the fact that for a fair comparison, the same prior knowledge about the data structure (time series) should be used. In all other aspects we applied the method as described in [11]. All MCMC simulations were divided into a burn-in and a sampling phase, where the length of the burn-in phase was chosen such that standard convergence criteria based on potential scale reduction factors [25] were met. The software implementations of all methods used in our study are available upon request. For lack of space, further details have to be relegated to the supplementary material.

To assess the network reconstruction accuracy, various criteria have been proposed in the literature. In the present study, we chose receiver-operator-characteristic (ROC) curves computed from the marginal posterior probabilities of the edges (and the ranking thereby induced). Owing to the large number of simulations – for each network and parameter setting the simulations were repeated on 25 (Figures 2a-c) or 5 (Figures 2d) independently generated time series – we summarized the performance by the area under the curve (AUC), ranging between 0.5 (expected random predictor) to 1.0 (perfect predictor). The results are shown in Figure 2 and suggest that the proposed cpBGe model tends to significantly outperform the competing models. A more detailed analysis with an

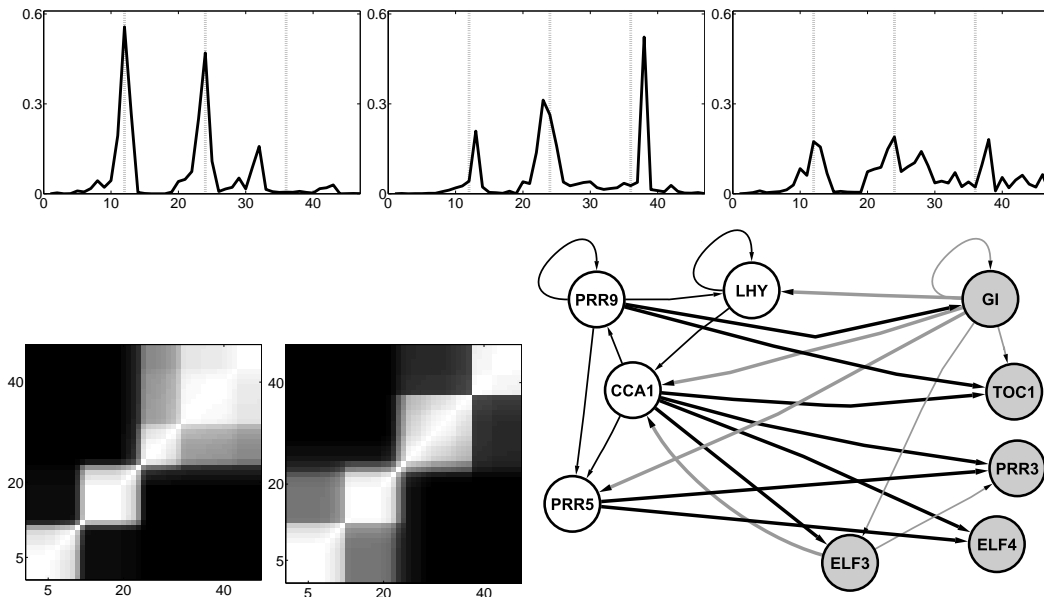

Figure 3: **Results on the Arabidopsis gene expression time series.** *Top panels:* Average posterior probability of a change-point (vertical axis) at a specific transition time plotted against the transition time (horizontal axis) for two selected circadian genes (left: LHY, centre: TOC1) and averaged over all 9 genes (right). The vertical dotted lines indicate the boundaries of the time series segments, which are related to different entrainment conditions and time intervals. *Bottom left and centre panels:* Co-allocation matrices for the two selected genes LHY and TOC1. The axes represent time. The grey shading indicates the posterior probability of two time points being assigned to the same mixture component, ranging from 0 (black) to 1 (white). *Bottom right panel:* Predicted regulatory network of nine circadian genes in *Arabidopsis thaliana*. Empty circles represent morning genes. Shaded circles represent evening genes. Edges indicate predicted interactions with a marginal posterior probability greater than 0.5.

investigation of how the signal-to-noise ratio and the autocorrelation parameters effect the relative performance of the methods has to be relegated to the supplementary material for lack of space.

## 4   Results on Arabidopsis gene expression time series

We have applied our method to microarray gene expression time series related to the study of circadian regulation in plants. *Arabidopsis thaliana* seedlings, grown under artificially controlled $T_e$-hour-light/$T_e$-hour-dark cycles, were transferred to constant light and harvested at 13 time points in $\tau$-hour intervals. From these seedlings, RNA was extracted and assayed on Affymetrix GeneChip oligonucleotide arrays. The data were background-corrected and normalized according to standard procedures[5], using the GeneSpring$^{©}$ software (Agilent Technologies). We combined four time series, which differed with respect to the pre-experiment entrainment condition and the time intervals: $T_e \in \{10h, 12h, 14h\}$, and $\tau \in \{2h, 4h\}$. The data, with detailed information about the experimental protocols, can be obtained from [27], [11], and [28]. We focused our analysis on 9 circadian genes[6] (i.e. genes involved in circadian regulation). We combined all four time series into a single set. The objective was to test whether the proposed cpBGe model would detect the different experimental phases. Since the gene expression values at the first time point of a time series segment have no relation with the expression values at the last time point of the preceding segment, the corresponding boundary time points were appropriately removed from the data[7]. This ensures that for all pairs of consecutive time points a proper conditional dependence relation determined by the nature of the regulatory cellular processes is given. The top panel of Figure 3 shows the marginal posterior

probability of a change-point for two selected genes (LHY and TOC1), and averaged over all genes. It is seen that the three concatenation points are clearly detected. There is a slight difference between the heights of the posterior probability peaks for LHY and TOC1. This behaviour is also captured by the co-allocation matrices in the bottom row of Figure 3. This deviation indicates that the two genes are effected by the changing experimental conditions (entrainment, time interval) in different ways and thus provides a useful tool for further exploratory analysis. The bottom right panel of Figure 3 shows the gene interaction network that is predicted when keeping all edges with marginal posterior probability above 0.5. There are two groups of genes. Empty circles in the figure represent morning genes (i.e. genes whose expression peaks in the morning), shaded circles represent evening genes (i.e. genes whose expression peaks in the evening). There are several directed edges pointing from the group of morning genes to the evening genes, mostly originating from gene CCA1. This result is consistent with the findings in [29], where the morning genes were found to activate the evening genes, with CCA1 being a central regulator. Our reconstructed network also contains edges pointing into the opposite direction, from the evening genes back to the morning genes. This finding is also consistent with [29], where the evening genes were found to inhibit the morning genes via a negative feedback loop. In the reconstructed network, the connectivity within the group of evening genes is sparser than within the group of morning genes. This finding is consistent with the fact that following the light-dark cycle entrainment, the experiments were carried out in constant-light condition, resulting in a higher activity of the morning genes overall. Within the group of evening genes, the reconstructed network contains an edge between GI and TOC1. This interaction has been confirmed in [30]. Hence while a proper evaluation of the reconstruction accuracy is currently unfeasible – like [6] and many related studies, we lack a gold-standard owing to the unknown nature of the true interaction network – our study suggests that the essential features of the reconstructed network are biologically plausible and consistent with the literature.

## 5  Discussion

We have proposed a continuous-valued non-stationary dynamic Bayesian network, which constitutes a non-stationary generalization of the BGe model. This complements the work of [6], where a non-stationary BDe model was proposed. We have argued that a flexible network *structure* can lead to practical and conceptual problems, and we therefore only allow the *parameters* to vary with time. We have presented a comparative evaluation of the network reconstruction accuracy on synthetic data. Note that such a study is missing from recent related studies on this topic, like [6] and [7], presumably because their overall network structure is not properly defined. Our findings suggest that the proposed non-stationary BGe model achieves a clear performance improvement over the classical stationary models BDe and BGe as well as over the non-linear/non-stationary models of [12] and [11]. The application of our model to gene expression time series from circadian clock-regulated genes in *Arabidopsis thaliana* has led to a plausible data segmentation, and the reconstructed network shows features that are consistent with the biological literature.

The proposed model is based on a multiple change-point process. This scheme provides the approximation of a non-linear regulation process by a piecewise linear process under the assumption that the temporal processes are sufficiently smooth. A straightforward modification would be the replacement of the change-point process by the allocation model of [13] and [11]. This modification would result in a fully-flexible mixture model, which is preferable if the smoothness assumption for the temporal processes is violated. It would also provide a non-linear Bayesian network for static rather than time series data. While the algorithmic implementation is straightforward, the increased complexity of the latent variable configuration space would introduce additional challenges for the mixing and convergence properties of the MCMC sampler. The development of more effective proposal moves, as well as a comparison with alternative non-linear Bayesian network models, like [31], is a promising subject for future research.

## Acknowledgements

Marco Grzegorczyk is supported by the Graduate School "Statistische Modellbildung" of the Department of Statistics, University of Dortmund. Dirk Husmeier is supported by the Scottish Government Rural and Environment Research and Analysis Directorate (RERAD).

## Footnotes

[1]Note that as opposed to [7], [6] partially addresses this issue via a prior distribution that discourages changes in the network structure.

[2]The conjugate prior is a normal-Wishart distribution. For the present study, we chose the hyperparameters of this distribution maximally uninformative subject to the regularity conditions discussed in [10].

[3]The score equivalence aspect of the BGe model is not required for DBNs, because edge reversals are not permissible. However, formulating our method in terms of the BGe score is advantageous when adapting the proposed framework to non-linear static Bayesian networks along the lines of [12].

[4]An MCMC algorithm based on Eq.(10) in [16] is computationally less efficient than when applied to static Bayesian networks or stationary DBNs, since the local scores would have to be re-computed every time the positions of the change-points change.

[5] We used RMA rather than GCRMA for reasons discussed in [26].

[6] These 9 circadian genes are LHY, TOC1, CCA1, ELF4, ELF3, GI, PRR9, PRR5, and PRR3.

[7] A proper mathematical treatment is given in Section 3 of the supplementary material.

# References

[1] N. Friedman, M. Linial, I. Nachman, and D. Pe'er. Using Bayesian networks to analyze expression data. *Journal of Computational Biology*, 7:601–620, 2000.

[2] K. Sachs, O. Perez, D. Pe'er, D. A. Lauffenburger, and G. P. Nolan. Protein-signaling networks derived from multiparameter single-cell data. *Science*, 308:523–529, 2005.

[3] V. A. Smith, J. Yu, T. V. Smulders, A. J. Hartemink, and E. D. Jarvi. Computational inference of neural information flow networks. *PLoS Computational Biology*, 2:1436–1449, 2006.

[4] M. Talih and N. Hengartner. Structural learning with time-varying components: Tracking the cross-section of financial time series. *Journal of the Royal Statistical Society B*, 67(3):321–341, 2005.

[5] X. Xuan and K. Murphy. Modeling changing dependency structure in multivariate time series. In Zoubin Ghahramani, editor, *Proceedings of the 24th Annual International Conference on Machine Learning (ICML 2007)*, pages 1055–1062. Omnipress, 2007.

[6] J. W. Robinson and A. J. Hartemink. Non-stationary dynamic Bayesian networks. In D. Koller, D. Schuurmans, Y. Bengio, and L. Bottou, editors, *Advances in Neural Information Processing Systems 21*, pages 1369–1376. Morgan Kaufmann Publishers, 2009.

[7] S. Lèbre. *Analyse de processus stochastiques pour la génomique : étude du modèle MTD et inférence de réseaux bayésiens dynamiques*. PhD thesis, Université d'Evry-Val-d'Essonne, 2008.

[8] D. Heckerman, D. Geiger, and D. M. Chickering. Learning Bayesian networks: The combination of knowledge and statistical data. *Machine Learning*, 20:245–274, 1995.

[9] C. Andrieu and A. Doucet. Joint Bayesian model selection and estimation of noisy sinusoids via reversible jump MCMC. *IEEE Transactions on Signal Processing*, 47(10):2667–2676, 1999.

[10] D. Geiger and D. Heckerman. Learning Gaussian networks. In *Proceedings of the Tenth Conference on Uncertainty in Artificial Intelligence*, pages 235–243, San Francisco, CA., 1994. Morgan Kaufmann.

[11] M. Grzegorczyk, D. Husmeier, K. Edwards, P. Ghazal, and A. Millar. Modelling non-stationary gene regulatory processes with a non-homogeneous Bayesian network and the allocation sampler. *Bioinformatics*, 24(18):2071–2078, 2008.

[12] Y. Ko, C. Zhai, and S.L. Rodriguez-Zas. Inference of gene pathways using Gaussian mixture models. In *BIBM International Conference on Bioinformatics and Biomedicine*, pages 362–367. Fremont, CA, 2007.

[13] A. Nobile and A.T. Fearnside. Bayesian finite mixtures with an unknown number of components: The allocation sampler. *Statistics and Computing*, 17(2):147–162, 2007.

[14] G. Schwarz. Estimating the dimension of a model. *Annals of Statistics*, 6:461–464, 1978.

[15] P. Green. Reversible jump Markov chain Monte Carlo computation and Bayesian model determination. *Biometrika*, 82:711–732, 1995.

[16] N. Friedman and D. Koller. Being Bayesian about network structure. *Machine Learning*, 50:95–126, 2003.

[17] P. Giudici and R. Castelo. Improving Markov chain Monte Carlo model search for data mining. *Machine Learning*, 50:127–158, 2003.

[18] D. Madigan and J. York. Bayesian graphical models for discrete data. *International Statistical Review*, 63:215–232, 1995.

[19] P. Fearnhead. Exact and efficient Bayesian inference for multiple changepoint problems. *Statistics and Computing*, 16:203–213, 2006.

[20] S. S. Shen-Orr, R. Milo, S. Mangan, and U. Alon. Network motifs in the transcriptional regulation network of *Escherichia coli*. *Nature Genetics*, 31:64–68, 2002.

[21] M. K. Dougherty, J. Muller, D. A. Ritt, M. Zhou, X. Z. Zhou, T. D. Copeland, T. P. Conrads, T. D. Veenstra, K. P. Lu, and D. K. Morrison. Regulation of Raf-1 by direct feedback phosphorylation. *Molecular Cell*, 17:215–224, 2005.

[22] A. J. Hartemink. *Principled Computational Methods for the Validation and Discovery of Genetic Regulatory Networks*. PhD thesis, MIT, 2001.

[23] A. P. Dempster, N. M. Laird, and D. B. Rubin. Maximum likelihood from incomplete data via the EM algorithm. *Journal of the Royal Statistical Society*, B39(1):1–38, 1977.

[24] I. T. Nabney. *NETLAB: Algorithms for Pattern Recognition*. Springer Verlag, New York, 2004.

[25] A. Gelman and D. B. Rubin. Inference from iterative simulation using multiple sequences. *Statistical Science*, 7:457–472, 1992.

[26] W.K. Lim, K. Wang, C. Lefebvre, and A. Califano. Comparative analysis of microarray normalization procedures: effects on reverse engineering gene networks. *Bioinformatics*, 23(13):i282–i288, 2007.

[27] K. D. Edwards, P. E. Anderson, A. Hall, N. S. Salathia, J. C.W. Locke, J. R. Lynn, M. Straume, J. Q. Smith, and A. J. Millar. Flowering locus C mediates natural variation in the high-temperature response of the Arabidopsis circadian clock. *The Plant Cell*, 18:639–650, 2006.

[28] T.C. Mockler, T.P. Michael, H.D. Priest, R. Shen, C.M. Sullivan, S.A. Givan, C. McEntee, S.A. Kay, and J. Chory. The diurnal project: Diurnal and circadian expression profiling, model-based pattern matching and promoter analysis. *Cold Spring Harbor Symposia on Quantitative Biology*, 72:353–363, 2007.

[29] C. R. McClung. Plant circadian rhythms. *Plant Cell*, 18:792–803, 2006.

[30] J.C.W. Locke, M.M. Southern, L. Kozma-Bognar, V. Hibberd, P.E. Brown, M.S. Turner, and A.J. Millar. Extension of a genetic network model by iterative experimentation and mathematical analysis. *Molecular Systems Biology*, 1:(online), 2005.

[31] S. Imoto, S. Kim, T. Goto, , S. Aburatani, K. Tashiro, Satoru Kuhara, and Satoru Miyano. Bayesian networks and nonparametric heteroscedastic regression for nonlinear modeling of genetic networks. *Journal of Bioinformatics and Computational Biology*, 1(2):231–252, 2003.

